# Reinforcement Learning for Call Admission Control and Routing in Integrated Service Networks

**Peter Marbach***
LIDS
MIT
Cambridge, MA, 02139
email: marbach@mit.edu

**Oliver Mihatsch**
Siemens AG
Corporate Technology, ZT IK 4
D-81730 Munich, Germany
email: oliver.mihatsch@
mchp.siemens.de

**Miriam Schulte**
Zentrum Mathematik
Technische Universität München
D-80290 Munich
Germany

**John N. Tsitsiklis**
LIDS
MIT
Cambridge, MA, 02139
email: jnt@mit.edu

## Abstract

In integrated service communication networks, an important problem is to exercise call admission control and routing so as to optimally use the network resources. This problem is naturally formulated as a dynamic programming problem, which, however, is too complex to be solved exactly. We use methods of reinforcement learning (RL), together with a decomposition approach, to find call admission control and routing policies. The performance of our policy for a network with approximately $10^{45}$ different feature configurations is compared with a commonly used heuristic policy.

## 1 Introduction

The call admission control and routing problem arises in the context where a telecommunication provider wants to sell its network resources to customers in order to maximize long term revenue. Customers are divided into different classes, called service types. Each service type is characterized by its bandwidth demand, its average call holding time and the immediate reward the network provider obtains, whenever a call of that service type is

accepted. The control actions for maximizing the long term revenue are to accept or reject new calls (*Call Admission Control*) and, if a call is accepted, to route the call appropriately through the network (*Routing*). The problem is naturally formulated as a dynamic programming problem, which, however, is too complex to be solved exactly. We use the methodology of reinforcement learning (RL) to approximate the value function of dynamic programming. Furthermore, we pursue a decomposition approach, where the network is viewed as consisting of link processes, each having its own value function. This has the advantage, that it allows a decentralized implementation of the training methods of RL and a decentralized implementation of the call admission control and routing policies. Our method learns call admission control and routing policies which outperform the commonly used heuristic "Open-Shortest-Path-First" (OSPF) policy.

In some earlier related work, we applied RL to the call admission problem for a single communication link in an integrated service environment. We found that in this case, RL methods performed as well, but no better than, well-designed heuristics. Compared with the single link problem, the addition of routing decisions makes the network problem more complex and good heuristics are not easy to derive.

## 2   Call Admission Control and Routing

We are given a telecommunication network consisting of a set of nodes $\mathcal{N} = \{1, ..., N\}$ and a set of links $\mathcal{L} = \{1, ..., L\}$, where link $l$ has a a total capacity of $B(l)$ units of bandwidth. We support a set $\mathcal{M} = \{1, ..., M\}$ of different service types, where a service type $m$ is characterized by its bandwidth demand $b(m)$, its average call holding time $1/\nu(m)$ (here we assume that the call holding times are exponentially distributed) and the immediate reward $c(m)$ we obtain, whenever we accept a call of that service type. A link can carry simultaneously any combination of calls, as long as the bandwidth used by these calls does not exceed the total bandwidth of the link (*Capacity Constraint*). When a new call of service type $m$ requests a connection between a node $i$ and a node $j$, we can either reject or accept that request (*Call Admission Control*). If we accept the call, we choose a route out of a list of predefined routes (*Routing*). The call then uses $b(m)$ units of bandwidth on each link along that route for the duration of the call. We can, therefore, only choose a route, which does not violate the capacity constraints of its links, if the call is accepted. Furthermore, if we accept the call, we obtain an immediate reward $c(m)$. The objective is to exercise call admission control and routing in such a way that the long term revenue obtained by accepting calls is maximized.

We can formulate the call admission control and routing problem using dynamic programming (e. g. Bertsekas, 1995). Events $\omega$ which incur state transitions, are arrivals of new calls and call terminations. The state $x_t$ at time t consists of a list for each route, indicating how many calls of each service type are currently using that route. The decision/control $u_t$ applied at the time $t$ of an arrival of a new call is to decide, whether to reject or accept the call, and, if the call is accepted, how to route it through the network. The objective is to learn a policy that assigns decisions to each state so as to

$$\text{maximize} \quad \left( J = E \left\{ \sum_{k=0}^{\infty} e^{-\beta t_k} g(x_{t_k}, \omega_k, u_{t_k}) \right\} \right)$$

where $E\{\cdot\}$ is the expectation operator, $t_k$ is the time when the $k$th event happens, $g(x_{t_k}, \omega_k, u_{t_k})$ is the immediate reward associated with the $k$th event, and $\beta$ is a discount factor that makes immediate rewards more valuable than future ones.

## 3  Reinforcement Learning Solution

RL methods solve optimal control (or dynamic programming) problems by learning good approximations to the optimal value function $J^*$, given by the solution to the Bellman optimality equation which takes the following form for the call admission control and routing problem

$$J^*(x) = E_\tau \left\{ e^{-\beta\tau} \right\} E_\omega \left\{ \max_{u \in U(x)} [g(x, \omega, u) + J^*(x')] \right\}$$

where $U(x)$ is the set of control actions available in the current state $x$, $\tau$ is the time when the first event $\omega$ occurs and $x'$ is the successor state. Note that $x'$ is a deterministic function of the current state $x$, the control $u$ and the event $\omega$.

RL uses a compact representation $\tilde{J}(\cdot, \theta)$ to learn and store an estimate of $J^*(\cdot)$. On each event, $\tilde{J}(\cdot, \theta)$ is both used to make decisions and to update the parameter vector $\theta$. In the call admission control and routing problem, one has only to choose a control action when a new call requests a connection. In such a case, $\tilde{J}(\cdot, \theta)$ is used to choose a control action according to the formula

$$u = \arg \max_{u \in U(x)} \left[ g(x, \omega, u) + \tilde{J}(x', \theta) \right] \tag{1}$$

This can be expressed in words as follows.

**Decision Making:** When a new call requests a connection, use $\tilde{J}(\cdot, \theta)$ to evaluate, for each permissible route, the successor state $x'$ we transit to, when we choose that route, and pick a route which maximizes that value. If the sum of the immediate reward and the value associated with this route is higher than the value of the current state, route the call over that route; otherwise reject the call.

Usually, RL uses a global feature extractor $f(x)$ to form an approximate compact representation of the state of the system, which forms the input to a function approximator $\tilde{J}(\cdot, \theta)$. Sutton's temporal difference (TD($\lambda$)) algorithms (Sutton, 1988) can then be used to train $\tilde{J}(\cdot, \theta)$ to learn an estimate of $J^*$. Using TD(0), the update at the $k$th event takes the following form

$$\theta_k = \theta_{k-1} + \gamma_k d_k \nabla_\theta \tilde{J}(f(x_{t_{k-1}}), \theta_{k-1})$$

where

$$
\begin{aligned}
d_k &= e^{-\beta(t_k - t_{k-1})} \left( g(x_{t_k}, \omega_k, u_{t_k}) + \tilde{J}(f(x_{t_k}), \theta_{k-1}) \right) \\
&\quad - \tilde{J}(f(x_{t_{k-1}}), \theta_{k-1})
\end{aligned}
$$

and where $\gamma_k$ is a small step size parameter and $u_{t_k}$ is the control action chosen according to the decision making rule described above.

Here we pursue an approach where we view the network as being composed of link processes. Furthermore, we decompose immediate rewards $g(x_{t_k}, \omega_k, u_{t_k})$ associated with the $k$th event, into link rewards $g^{(l)}(x_{t_k}, \omega_k, u_{t_k})$ such that

$$g(x_{t_k}, \omega_k, u_{t_k}) = \sum_{l=1}^{L} g^{(l)}(x_{t_k}, \omega_k, u_{t_k})$$

We then define, for each link $l$, a value function $\tilde{J}^{(l)}(f^{(l)}(x), \theta^{(l)})$, which is interpreted as an estimate of the discounted long term revenue associated with that link. Here, $f^{(l)}$ defines a local feature, which forms the input to the value function associated with link $l$. To obtain

an approximation of $J^*(x)$, the functions $\tilde{J}^{(l)}(f^{(l)}(x), \theta^{(l)})$ are combined as follows

$$\sum_{l=1}^{L} \tilde{J}^{(l)}(f^{(l)}(x), \theta^{(l)}).$$

At each event, we update the parameter vector $\theta^{(l)}$ of link $l$, only if the event is associated with the link. Events associated with a link $l$ are arrivals of new calls which are potentially routed over link $l$ and termination of calls which were routed over the link $l$. The update rule of the parameter vector $\theta^{(l)}$ is very similar to the TD(0) algorithm described above

$$\theta_k^{(l)} = \theta_{k-1}^{(l)} + \gamma_k^{(l)} d_k^{(l)} \nabla_{\theta^{(l)}} \tilde{J}^{(l)}(f^{(l)}(x_{t_{k-1}}), \theta_{k-1}^{(l)}) \tag{2}$$

where

$$
\begin{aligned}
d_k^{(l)} = {}& e^{-\beta(t_k^{(l)} - t_{k-1}^{(l)})} \left( g^{(l)}(x_{t_k^{(l)}}, \omega_k^{(l)}, u_{t_k^{(l)}}) + \tilde{J}^{(l)}(f^{(l)}(x_{t_k^{(l)}}), \theta_{k-1}^{(l)}) \right) \\
& - \tilde{J}^{(l)}(f^{(l)}(x_{t_{k-1}^{(l)}}), \theta_{k-1}^{(l)})
\end{aligned}
\tag{3}
$$

and where $\gamma_k^{(l)}$ is a small step size parameter and $t_k^{(l)}$ is the time when the $k$th event $\omega_k^{(l)}$ associated with link $l$ occurs. Whenever a new call of a service of type $m$ is routed over a route $r$ which contains the link $l$, the immediate reward $g^{(l)}$ associated with the link $l$ is equal to $c(m)/\#r$, where $\#r$ is the number of links along the route $r$. For all other events, the immediate reward associated with link $l$ is equal to 0.

The advantage of this decomposition approach is that it allows decentralized training and decentralized decision making. Furthermore, we observed that this decomposition approach leads to much shorter training times for obtaining an approximation for $J^*$ than the approach without decomposition. All these features become very important if one considers applying methods of RL to large integrated service networks supporting a fair number of different service types.

We use exploration to obtain the states at which we update the parameter vector $\theta$. At each state, with probability $p = 0.5$, we apply a random action, instead of the action recommended by the current value function, to generate the next state in our training trajectory. However, the action $u_{t_k^{(l)}}$, that is used in the update rule (3), is still the one chosen according to the rule given in (1). Exploration during the training significantly improved the performance of the policy.

Table 1: Service Types.

| SERVICE TYPE $m$ | 1 | 2 | 3 |
|---|---|---|---|
| BANDWIDTH DEMAND $b(m)$ | 1 | 3 | 5 |
| AVERAGE HOLDING TIME $1/\nu(m)$ | 10 | 10 | 2 |
| IMMEDIATE REWARD $c(m)$ | 1 | 2 | 50 |

# 4  Experimental Results

In this section, we present experimental results obtained for the case of an integrated service network consisting of 4 nodes and 12 unidirectional links. There are two different classes of links with a total capacity of 60 and 120 units of bandwidth, respectively (indicated by thick and thin arrows in Figure 1). We assume a set $\mathcal{M} = \{1, 2, 3\}$ of three different service types. The corresponding bandwidth demands, average holding times and immediate

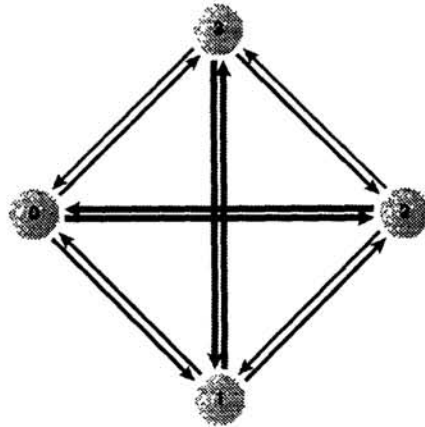

Figure 1: Telecommunication Network Consisting of 4 Nodes and 12 Unidirectional Links.

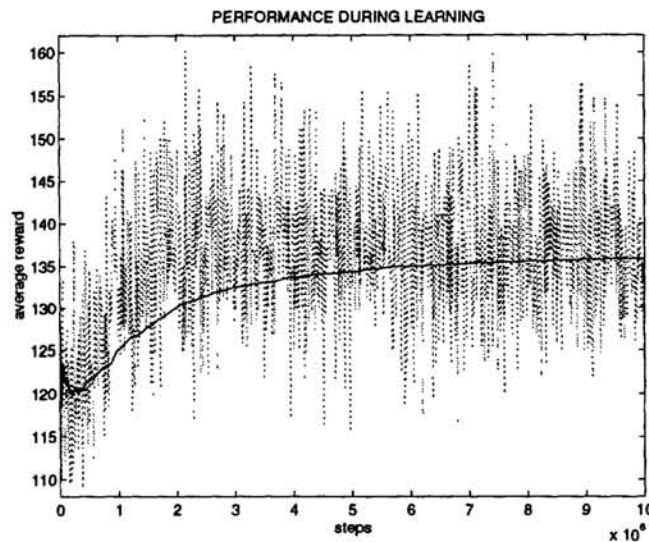

Figure 2: Average Reward per Time Unit During the Whole Training Phase of $10^7$ Steps (Solid) and During Shorter Time Windows of $10^5$ Steps (Dashed).

rewards are given in Table 1. Call arrivals are modeled as independent Poisson processes, with a separate mean for each pair of source and destination nodes and each service type. Furthermore, for each source and destination node pair, the list of possible routes consists of three entries: the direct path and the two alternative 2-hop-routes.

We compare the policy obtained through RL with the commonly used heuristic OSPF (Open Shortest Path First). For every pair of source and destination nodes, OSPF orders the list of predefined routes. When a new call arrives, it is routed along the first route in the corresponding list, that does not violate the capacity constraint; if no such a route exists, the call is rejected. We use the average reward per unit time as performance measure to compare the two policies.

For the RL approach, we use a quadratic approximator, which is linear with respect to the parameters $\theta^{(l)}$, as a compact representation of $\tilde{J}^{(l)}$. Other approximation architectures were tried, but we found that the quadratic gave the best results with respect to both the speed of convergence and the final performance. As inputs to the compact representation

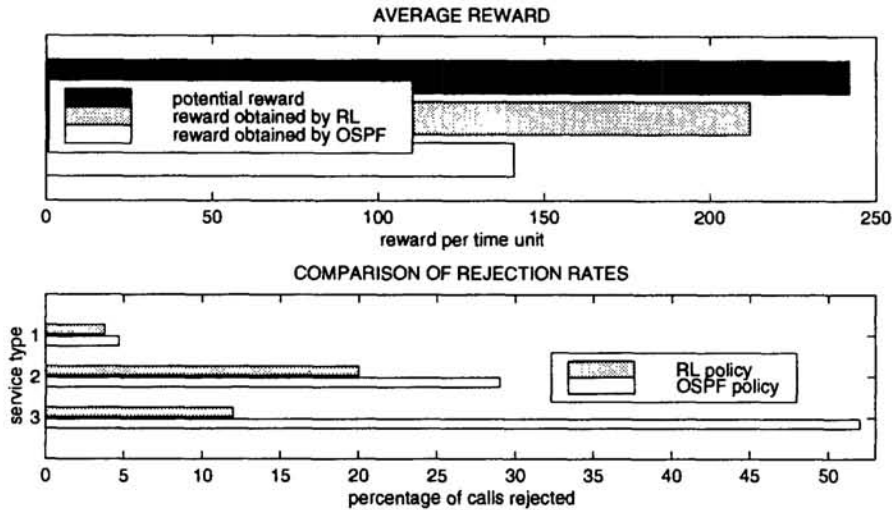

Figure 3: Comparison of the Average Rewards and Rejection Rates of the RL and OSPF Policies.

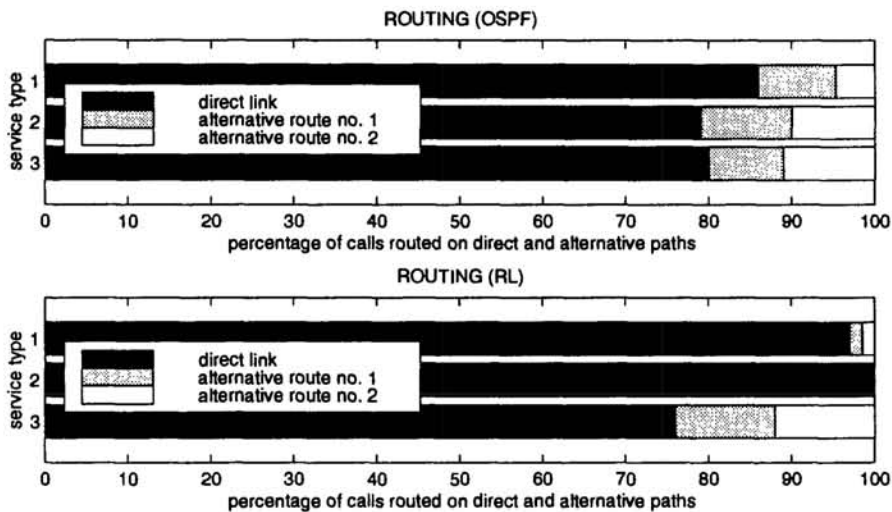

Figure 4: Comparison of the Routing Behaviour of the RL and OSPF Policies.

$\tilde{J}^{(l)}$, we use a set of local features, which we chose to be the number of ongoing calls of each service type on link $l$. For the 4-node network, there are approximately $1.6 \cdot 10^{45}$ different feature configurations. Note that the total number of possible states is even higher.

The results of the case studies are given in in Figure 2 (Training Phase), Figure 3 (Performance) and Figure 4 (Routing Behaviour). We give here a summary of the results.

**Training Phase:** Figure 2 shows the average reward of the RL policy as a function of the training steps. Although the average reward increases during the training, it does not exceed 141, the average reward of the heuristic OSPF. This is due to the high amount of exploration in the training phase.

**Performance Comparison:** The policy obtained through RL gives an average reward of 212, which as about 50% higher than the one of 141 achieved by OSPF. Furthermore, the RL policy reduces the number of rejected calls for all service types. The most significant reduction is achieved for calls of service type 3, the service type, which has the highest

immediate reward. Figure 3 also shows that the average reward of the RL policy is close to the potential average reward of 242, which is the average reward we would obtain if all calls were accepted. This leaves us to believe that the RL policy is close to optimal. Figure 4 compares the routing behaviour of the RL control policy and OSPF. While OSPF routes about $15\% - 20\%$ of all calls along one of the alternative 2-hop-routes, the RL policy almost uses alternative routes for calls of type 3 (about 25%) and routes calls of the other two service types almost exclusively over the direct route. This indicates, that the RL policy uses a routing scheme, which avoids 2-hop-routes for calls of service type 1 and 2, and which allows us to use network resources more efficiently.

## 5  Conclusion

The call admission control and routing problem for integrated service networks is naturally formulated as a dynamic programming problem, albeit one with a very large state space. Traditional dynamic programming methods are computationally infeasible for such large scale problems. We use reinforcement learning, based on Sutton's (1988) $TD(0)$, combined with a decomposition approach, which views the network as consisting of link processes. This decomposition has the advantage that it allows decentralized decision making and decentralized training, which reduces significantly the time of the training phase. We presented a solution for an example network with about $10^{45}$ different feature configurations. Our RL policy clearly outperforms the commonly used heuristic OSPF. Besides the game of backgammon (Tesauro, 1992), the elevator scheduling (Crites & Barto, 1996), the jop-shop scheduling (Zhang & Dietterich, 1996) and the dynamic channel allocation (Singh & Bertsekas, 1997), this is another successful application of RL to a large-scale dynamic programming problem for which a good heuristic is hard to find.

## References

Bertsekas, D. P. (1995) *Dynamic Programming and Optimal Control*. Athena Scientific, Belmont, MA.

Crites, R. H., Barto, A. G. (1996) Improving elevator performance using reinforcement learning. In D. S. Touretzky, M. C. Mozer and M. E. Hasselmo (eds.), *Advances in Neural Information Processing Systems 8*, pp. 1017–1023. Cambridge, MA: MIT Press.

Singh, S., Bertsekas, D. P. (1997) Reinforcement learning for dynamic channel allocation in cellular telephone systems. To appear in *Advances in Neural Information Processing Systems 9*, Cambridge, MA: MIT Press.

Sutton, R. S. (1988) Learning to predict by the method of temporal differences. *Machine Learning*, 3:9–44.

Tesauro, G. J. (1992) Practical issues in temporal difference learning. *Machine Learning*, 8(3/4):257–277.

Zhang, W., Dietterich, T. G. (1996) High performance job-shop scheduling with a time-delay $TD(\lambda)$ network. In D. S. Touretzky, M. C. Mozer and M. E. Hasselmo (eds.), *Advances in Neural Information Processing Systems 8*, pp. 1024–1030. Cambridge, MA: MIT Press.
